# Information Dynamics and Emergent Computation in Recurrent Circuits of Spiking Neurons

**Thomas Natschläger, Wolfgang Maass**
Institute for Theoretical Computer Science
Technische Universitaet Graz
A-8010 Graz, Austria
{tnatschl, maass}@igi.tugraz.at

## Abstract

We employ an efficient method using Bayesian and linear classifiers for analyzing the dynamics of information in high-dimensional states of generic cortical microcircuit models. It is shown that such recurrent circuits of spiking neurons have an inherent capability to carry out rapid computations on complex spike patterns, merging information contained in the order of spike arrival with previously acquired context information.

## 1   Introduction

Common analytical tools of computational complexity theory cannot be applied to recurrent circuits with complex dynamic components, such as biologically realistic neuron models and dynamic synapses. In this article we explore the capability of information theoretic concepts to throw light on emergent computations in recurrent circuit of spiking neurons. This approach is attractive since it may potentially provide a solid mathematical basis for understanding such computations. But it is methodologically difficult because of systematic errors caused by under-sampling problems that are ubiquitous even in extensive computer simulations of relatively small circuits. Previous work on these methodological problems had focused on estimating the information in spike trains, i.e. temporally extended protocols of the activity of one or a few neurons. In contrast to that this paper addresses methods for estimating the information that is instantly available to a neuron that has synaptic connections to a large number of neurons.

We will define the specific circuit model used for our study in section 2 (although the methods that we apply appear to be useful for to a much wider class of analog and digital recurrent circuits). The combination of information theoretic methods with methods from machine learning that we employ is discussed in section 3. The results of applications of these methods to the analysis of the distribution and dynamics of information in a generic recurrent circuit of spiking neurons are presented in section 4. Applications of these methods to the analysis of emergent computations are discussed in section 5.

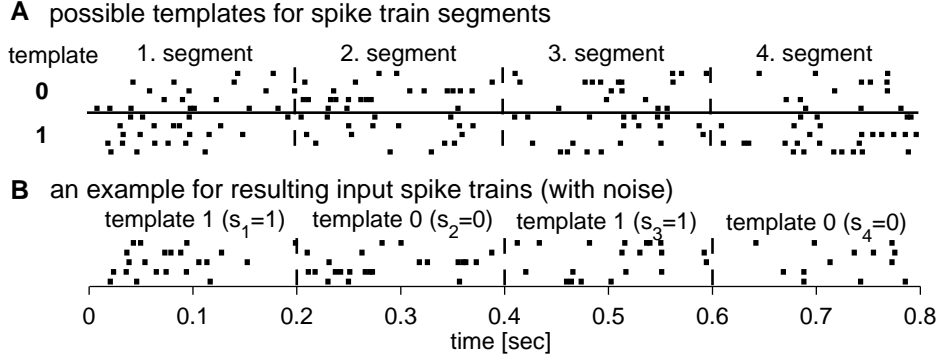

**A** possible templates for spike train segments

**B** an example for resulting input spike trains (with noise)

Figure 1: Input distribution used throughout the paper. Each input consists of 5 spike trains of length 800 ms generated from 4 segments of length 200 ms each. **A** For each segment 2 templates 0 and 1 were generated randomly (Poisson spike trains with a frequency of 20 Hz). **B** The actual input spike trains were generated by choosing randomly for each segment $i$, $i = 1, \ldots, 4$, one of the two associated templates ($s_i = 0$ or $s_i = 1$), and then generating a noisy version by moving each spike by an amount drawn from a Gaussian distribution with mean 0 and SD 4 ms.

## 2 Our study case: A Generic Neural Microcircuit Model

As our study case for analyzing information in high-dimensional circuit states we used a randomly connected circuit with sparse, primarily local connectivity consisting of 800 leaky integrate-and-fire (I&F) neurons, 20% of which were randomly chosen to be inhibitory. The 800 neurons of the circuit were arranged on two $20 \times 20$ layers L1 and L2. Circuit inputs consisting of 5 spike trains were injected into a randomly chosen subset of neurons in layer L1 (the connection probability was set to 0.25 for each of the 5 input channels and each neuron in layer L1). We modeled the (short term) dynamics of synapses according to the model proposed in [1], with the synaptic parameters $U$ (use), $D$ (time constant for depression), $F$ (time constant for facilitation) randomly chosen from Gaussian distributions that model empirical data for such connections. Parameters of neurons and synapses were chosen as in [2] to fit data from microcircuits in rat somatosensory cortex (based on [3] and [1]).

Since neural microcircuits in the nervous system often receive salient input in the form of spatio-temporal firing patterns (e.g. from arrays of sensory neurons, or from other brain areas), we have concentrated on circuit inputs of this type. Such firing pattern could for example represent visual information received during a saccade, or the neural representation of a phoneme or syllable in auditory cortex. Information dynamics and emergent computation in recurrent circuits of spiking neurons were investigated for input streams over 800 ms consisting of sequences of *noisy* versions of 4 of such firing patterns. We restricted our analysis to the case where in each of the four 200 ms segments one of two template patterns is possible, see Fig. 1. In the following we write $s_i = 1$ ($s_i = 0$) if a noisy version of template 1 (0) is used in the $i$-th time segment of the circuit input.

Fig. 2 shows the response of a circuit of spiking neurons (drawn from the distribution specified above) to the input stream exhibited in Fig. 1B. Each frame in Fig. 2 shows the current firing activity of one layer of the circuit at a particular point $t$ in time. Since in such rather small circuit (compared for example with the estimated $10^5$ neurons below a mm$^2$ of cortical surface) very few neurons fire at any given ms, we have replaced each spike by a pulse whose amplitude decays exponentially with a time constant of 30 ms. This models the impact of a spike on the membrane potential of a generic postsynaptic neuron. The resulting vector $r(t) = \langle r_1(t), \ldots, r_{800}(t) \rangle$ consisting of 800 analog values from the

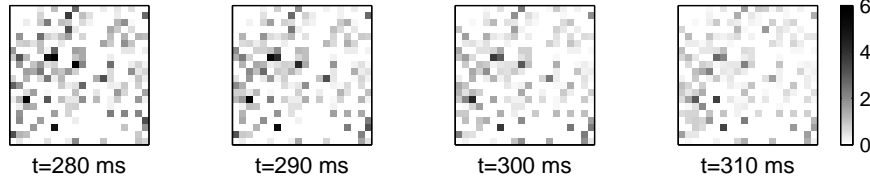

t=280 ms  t=290 ms  t=300 ms  t=310 ms

Figure 2: Snapshots of the first 400 components of the circuit state $r(t)$ (corresponding to the neurons in the layer L1) at various times $t$ for the input shown at the bottom of fig. 1. Black denotes high activity, white no activity. A spike at time $t_s \leq t$ adds a value of $\exp(-(t - t_s)/(30ms))$ to the corresponding component of the state $r(t)$.

800 neurons in the circuit is exactly the "liquid state" of the circuit at time $t$ in the context of the abstract computational model introduced in [2]. In the subsequent sections we will analyze the temporal dynamics of the information contained in these momentary circuit states $r(t)$.[1]

## 3  Methods for Analyzing the Information contained in Circuit States

The mutual information $MI(X, R)$ between two random variables $X$ and $R$ can be defined by $MI(X, R) = H(X) - H(X|R)$, where $H(X) = -\sum_{x \in Range(X)} p(x) \log p(x)$ is the entropy of $X$, and $H(X|R)$ is the expected value (with regard to $R$) of the conditional entropy of $X$ given $R$, see e.g. [4]. It is well known that empirical estimates of the entropy tend to underestimate the true entropy of a random variable (see e.g. [5, 6]). Hence in situations where the true value of $H(X)$ is known (as is typically the case in neuroscience applications where $X$ represents the stimulus, whose distribution is controlled by the experimentalist), the generic underestimate of $H(X|R)$ yield a generic overestimate of the mutual information $MI(X, R) = H(X) - H(X|R)$ for finite sample sizes. This under-sampling effect has been addressed in a number of studies (see e.g. [7], [8] and [9] and the references therein), and has turned out to be a serious obstacle for a wide-spread application of information theoretic methods to the analysis of neural computation. The seriousness of this problem becomes obvious from results achieved for our study case of a generic neural microcircuit shown in Fig. 3A. The dashed line shows the dependence of "raw" estimates $MI_{raw}$ of the mutual information $MI(s_2, R)$ on the sample size[2] $N$, which ranges here from $10^3$ to $2 \cdot 10^5$. The raw estimate of $MI(s_2, R)$ results from a direct application of the definition of $MI$ to the observed occupancy frequencies for a discrete set of bins[3], where $R$ consists here of just $d = 5$ or $d = 10$ components of the 800-dimensional circuit state $r(t)$ for $t = 660$ ms, and $s_2$ is the bit encoded by the second input segment. For more components $d$ of the current circuit state $r(t)$, e.g. for estimating the mutual information $MI(s_2, R)$ between the preceding circuit input $s_2$ and the current firing activity in a sub-circuit consisting of $d = 20$ or more neurons, even sample sizes beyond $10^6$ are likely to severely overestimate this mutual information.

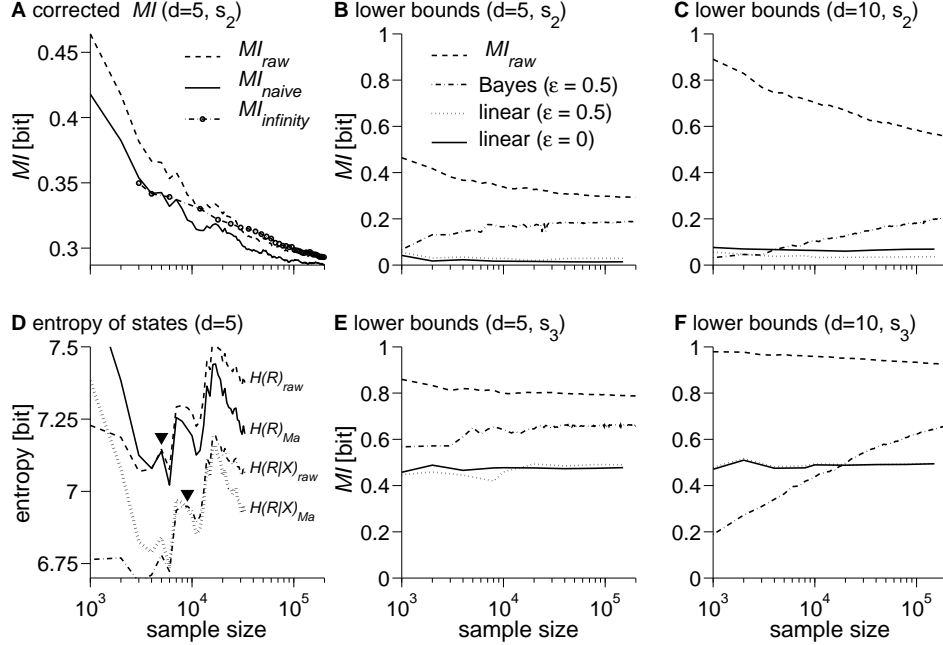

Figure 3: Estimated mutual information depends on sample size. In all panels $d$ denotes the number of components of the circuit state $r(t)$ at time $t = 660$ ms (or equivalently the number of neurons considered). **A** Dependence of the "raw" estimate $MI_{raw}$ and two corrected estimates $MI_{naive}$ and $MI_{infinity}$ of the mutual information $MI(s_2, R)$ (see text). **B** Lower bounds $MI(s_2, h(R))$ for the mutual information obtained via classifiers $h$ which are trained to predict the actual value of $s_2$ given the circuit state $r(t)$. Results are shown for a) an empirical Bayes classifier (discretization $\epsilon = 0.5$, see footnote 3 and 5), b) a linear classifier trained on the discrete ($\epsilon = 0.5$) data and c) for a linear classifier trained on the analog data ($\epsilon = 0$). In the case of the Bayes classifier $MI(s_2, h(R))$ was estimated by employing a leave-one-out procedure (which is computationally efficient for a Bayes classifier), whereas for the linear classifiers a test set of size $5 \cdot 10^4$ was used (hence no results beyond a sample size of $1.5 \cdot 10^5$). **C** Same as B but for $d = 10$. **D** Estimates of the entropies $H(R)$ and $H(R|X)$. The "raw" estimates are compared with the corresponding Ma-bounds (see text). The filled triangle marks the sample size from which on the Ma-bound is below the raw estimate. **E** Same as B but for $MI(s_3, h(R))$. **F** Same as E but for $d = 10$.

Several methods for correcting this bias towards overestimation of $MI$ have been suggested in the literature. In section 3.1 of [7] it is proposed to subtract one of two possible bias correction terms $B_{naive}$ and $B_{full}$ from the raw estimate $MI_{raw}$ of the mutual information. The effect of subtracting $B_{naive}$ is shown for $d = 5$ components of $r(t)$ in Fig. 3A. This correction is too optimistic for these applications, since the corrected estimate $MI_{naive} = MI_{raw} - B_{naive}$ at small sample sizes (e.g. $10^4$) is still substantially larger than the raw estimate $MI_{raw}$ at large sample sizes (e.g. $10^5$). The subtraction of the second proposed term $B_{full}$ is not applicable in our situation because it yields for$MI_{full} = MI_{raw} - B_{full}$ values lower than zero for all considered sample sizes. The reason is, that $B_{full}$ is proportional to the quotient "number of possible response bins" / $N$ and the number of possible response bins is in the order of $30^{10}$ in this example. Another way to correct $MI_{raw}$ is proposed in [10]. This approach is based on a series expansion of $MI$ in $1/N$ [6] and is effectively a method to get an empirical estimate $MI_{infinity}$ of the mutual information for infinite sample size ($N \rightarrow \infty$). It can be seen in Fig. 3A that for

moderate sample sizes $MI_{infinity}$ also yields too optimistic estimates for $MI$.

Another method for dealing with generic overestimates of $MI$ has been proposed in [10]. This method it based on the equation $MI(X, R) = H(R) - H(R|X)$ and compares the raw estimates of $H(R)$ and $H(R|X)$ with the so-called Ma-bounds, and suggests to judge raw estimates of $H(R)$ and $H(R|X)$, and hence raw estimates of $MI(X, R) = H(R) - H(R|X)$, as being trustworthy as soon as the sample size is so large that the corresponding Ma-bounds (which are conjectured to be less affected by undersampling) assume values below the raw estimates of $H(R)$ and $H(R|X)$. According to this criterion a sample size of $9 \cdot 10^3$ would be sufficient in the case of 5-neuron subcircuits (i.e., $d = 5$ components of $r(t)$), c.f. Fig. 3D.[4] However, Fig. 3A shows that the raw estimate $MI_{raw}$ is still too high for $N = 9 \cdot 10^3$ since $MI_{raw}$ assumes a substantially smaller value at $N = 2 \cdot 10^5$.

In view of this unreliability of – even corrected – estimates for the mutual information we have employed standard methods from machine learning in order to derive *lower bounds* for the $MI$ (see for example [8] and [9] for references to preceding related work). This method is computationally feasible and yields with not too large sample sizes reliable lower bounds for the $MI$ even for large numbers of components of the circuit state. In fact, we will apply it in sections 4 and 5 even to the full 800-component circuit state $r(t)$. This method is quite simple. According to the data processing inequality [4] one has $MI(X, R) \geq MI(X, h(R))$ for *any* function $h$. Obviously $MI(X, h(R))$ is easier to estimate than $MI(X, R)$ if the dimension of $h(R)$ is substantially lower than that of $R$, especially if $h(R)$ assumes just a few discrete values. Furthermore the difference between $MI(X, R)$ and $MI(X, h(R))$ is minimal if $h(R)$ throws away only that information in $R$ that is not relevant for predicting the value of $X$. Hence it makes sense to use as $h$ a predictor or classifier that has been trained to predict the current value of $X$. Similar approaches for estimating a lower bound were motivated by the idea of predicting the stimulus $(X)$ given the neural response $(R)$ (see [8], [9] and the references therein). To get an unbiased estimate for $MI(X, h(R))$ one has to make sure that $MI(X, h(R))$ is estimated on data which have not been used for the training of $h$. To make the best use of the data one can alternatively use cross-validation or even leave-one-out (see [11]) to estimate $MI(X, h(R))$. Fig. 3B, 3C, 3E, and 3F show for 3 different predictors $h$ how the resulting lower bounds for the $MI$ depend on the sample size $N$.

It is noteworthy that the lower bounds $MI(X, h(R))$ derived with the empirical Bayes classifier[5] increase significantly with the sample size[6] and converge quite well to the upper bounds $MI_{raw}(X, R)$. This reflects the fact that the estimated joint probability density between $X$ and $R$ gets more and more accurate. Furthermore the computationally less demanding[7] use of linear classifiers $h$ also yields significant lower bounds for $MI(X, R)$, especially if the true value of $MI(X, R)$ is not too small. In our application this does not even require high numerical precision, since a coarse binning (see footnote 3) of the analog components of $r(t)$ suffices, see Fig. 3 B,C,E,F. All estimates of $MI(X, R)$ in

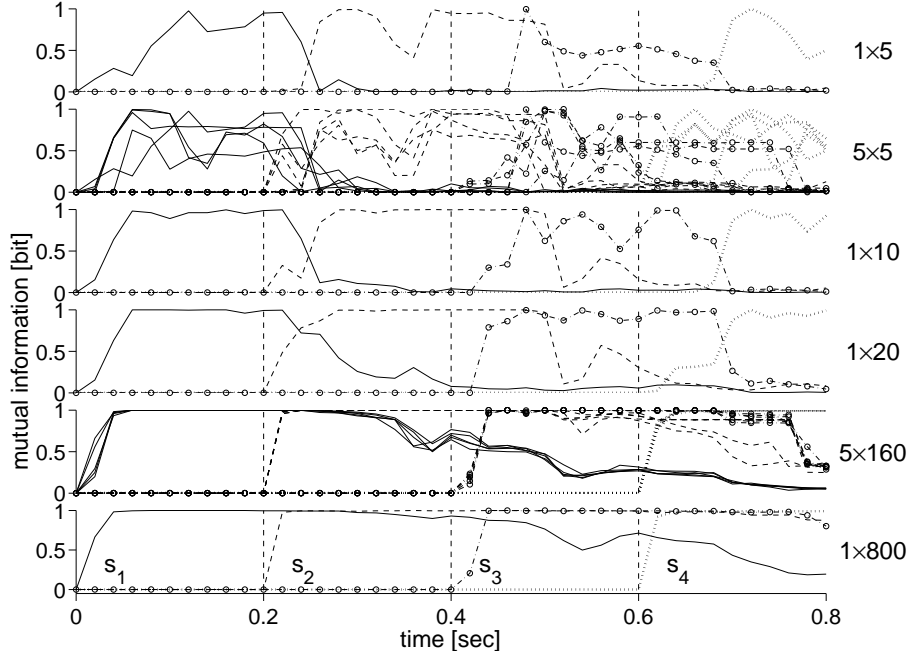

Figure 4: Information in subset of neurons. Shown are lower bounds for mutual information $MI(s_i, h(R))$ obtained with a linear classifier $h$ operating on $d$ components of the circuit state $\boldsymbol{r}(t)$. The numbers $a \times d$ to the right of each panel specify the number of components $d$ used by the linear classifier and for how many different choices $a$ of such subsets of size $d$ the results are plotted in that panel.

the subsequent sections are lower bounds $MI(X, h(R))$ computed via linear classifiers $h$. These types of lower bounds for $MI(X, R)$ are of particular interest from the point of view of neural computation, since a linear classifier can in principle be approximated by a neuron that is trained (for example by a suitable variation of the perceptron learning rule) to extract information about $X$ from the current circuit state $R$. Hence a high value of a lower bound $MI(X, h(R))$ for such $h$ shows not only that information about $X$ is present in the current circuit state $R$, but also that this information is in principle accessible for other neurons.

## 4 Distribution and Dynamics of Information in Circuit States

We have applied the method of estimating lower bounds for mutual information via linear classifiers described in the preceding section to analyze the spatial distribution and temporal dynamics of information for our study case described in section 2. Fig. 4 shows the temporal dynamics of information (estimated every 20ms as described in section 3) about input bits $s_i$ (encoded as described in section 2) for different components of the circuit state $\boldsymbol{r}(t)$ corresponding to different randomly drawn subsets of neurons in the circuit. One sees that even subsets of just 5 neurons absorb substantial information about the input bits $s_i$, however with a rather slow onset of the information uptake at the beginning of a segment and little memory retention when this information is overwritten by the next input segment. By merging the information from different subsets of neurons the uptake of new information gets faster and the memory retention grows. Note that for large sets of neurons (160 and 800) the information about each input bit $s_i$ jumps up to its maximal value right at the *beginning* of the corresponding $i$th segment of the input trains.

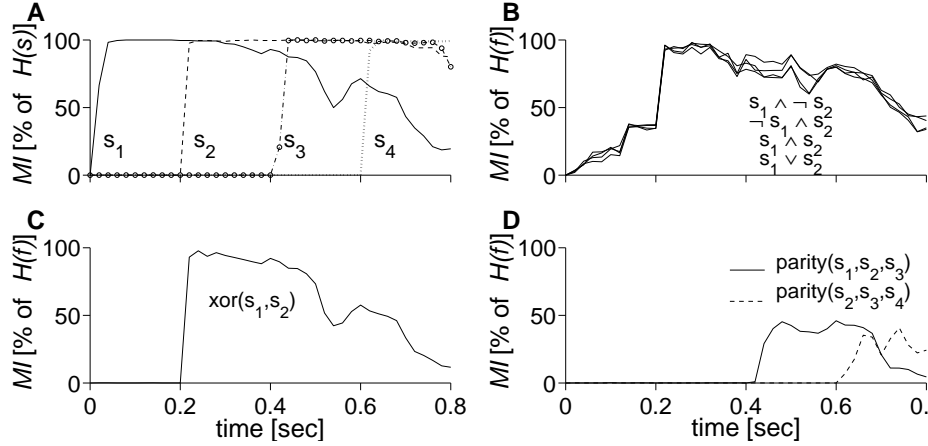

Figure 5: Emergent computations. **A** Dynamics of information about input bits as in the bottom row of Fig. 4. $H(s)$ denotes the entropy of a segment $s_i$ (which is 1 bit for $i = 1, 2, 3, 4$). **B, C, D** Lower bounds for the mutual information $MI(f, h(R))$ for various Boolean functions $f(s_1, \ldots, s_4)$ obtained with a linear classifier $h$ operating on the full 800-component circuit state $R = \boldsymbol{r}(t)$. $H(f)$ denotes the entropy of a Boolean function $f(s_1, \ldots, s_4)$ if the $s_i$ are independently uniformly drawn from $\{0, 1\}$.

## 5 Emergent Computation in Recurrent Circuits of Spiking Neurons

In this section we apply the same method to analyze the mutual information between the current circuit state and the target outputs of various computations on the information contained in the sequence of spatio-temporal spike patterns in the input stream to the circuit. This provides an interesting new method for analyzing neural computation, rather than just neural communication and coding. There exist 16 different Boolean functions $f(s_1, s_2)$ that depend just on the first two of the 4 bits $s_1, \ldots, s_4$. Fig. 5B,C shows that all these Boolean functions $f$ are autonomously computed by the circuit, in the sense that the current circuit state contains high mutual information with the target output $f(s_1, s_2)$ of this function $f$. Furthermore the information about the result $f(s_1, s_2)$ of this computation can be extracted *linearly* from the current circuit state $\boldsymbol{r}(t)$ (in spite of the fact that the computation of $f(s_1, s_2)$ from the spike patterns in the input requires highly nonlinear computational operations). This is shown in Fig. 5B and 5C for those 5 Boolean functions of 2 variables that are nontrivial in the sense that their output really depends on both input variables. There exist 5 other Boolean functions which are nontrivial in this sense, which are just the negations of the 5 Boolean functions shown (and for which the mutual information analysis therefore yields exactly the same result). In Fig. 5D corresponding results are shown for parity functions that depend on three of the 4 bits $s_1, s_2, s_3, s_4$. These Boolean functions are the most difficult ones to compute in the sense that knowledge of just 1 or 2 of their input bits does not give any advantage in guessing the output bit.

One noteworthy feature in all these emergent computations is that information about the result of the computation is already present in the current circuit state long before the complete spatio-temporal input patterns that encode the relevant input bits have been received by the circuit. In fact, the computation of $f(s_1, s_2)$ automatically just uses the temporal order of the first spikes in the pattern encoding $s_2$, and merges information contained in the order of these spikes with the "context" defined by the preceding input pattern. In this way the circuit automatically completes an ultra-rapid computation within just 20 ms of the beginning of the second pattern $s_2$. The existence of such ultra-rapid neural computations has previously already been inferred [12] but models that could explain the possibility of such ultra-rapid computations on the basis of generic models for recurrent neural microcircuits

have been missing.

## 6 Discussion

We have analyzed the dynamics of information in high-dimensional circuit states of a generic neural microcircuit model. We have focused on that information which can be extracted by a linear classifier (a linear classifier may be viewed as a coarse model for the classification capability of a biological neuron). This approach also has the advantage that significant lower bounds for the information content of high-dimensional circuit states can already be achieved for relatively small sample sizes. Our results show that information about current and preceding circuit inputs is spread throughout the circuit in a rather uniform manner. Furthermore our results show that a generic neural microcircuit model has inherent capabilities to process new input in the context of other information that arrived several hundred ms ago, and that information about the outputs of numerous potentially interesting target functions automatically accumulates in the current circuit state. Such emergent computation in circuits of spiking neurons is extremely fast, and therefore provides an interesting alternative to models based on special-purpose constructions for explaining empirically observed [12] ultra-rapid computations in neural systems.

The method for analyzing information contained in high-dimensional circuit states that we have explored in this article for a generic neural microcircuit model should also be applicable to biological data from multi-unit recordings, $fMRI$ etc., since significant lower bounds for mutual information were achieved in our study case already for sample sizes in the range of a few hundred (see Fig. 3). In this way one could get insight into the dynamics of information and emergent computations in biological neural systems.

**Acknowledgement:** We would like to thank Henry Markram for inspiring discussions. This research was partially supported by the Austrian Science Fund (FWF), project # P15386.

## Footnotes

[1]One should note that these circuit states do not reflect the complete current state of the underlying dynamical system, only those parts of the state of the dynamical system that are in principle "visible" for neurons outside the circuit. The current values of the membrane potential of neurons in the circuit and the current values of internal variables of dynamic synapses of the circuit are not visible in this sense.

[2]In our case the sample size $N$ refers to the number of computer simulations of the circuit response to new drawings of circuit inputs, with new drawings of temporal jitter in the input spike trains and initial conditions of the neurons in the circuit.

[3] For direct estimates of the $MI$ the analog value of each component of the circuit state $r(t)$ has to be divided into discrete bins. We first linearly transformed each component of $r(t)$ such that it has zeros mean and variance $\sigma^2 = 1.0$. The transformed components are then binned with a resolution of $\epsilon = 0.5$. This means that there are four bins in the range $\pm\sigma$.

[4] These kind of results depend on a division of the space of circuit states into subspaces, which is required for the calculation of the Ma-bound. In our case we have chosen the subspaces such that the frequency counts of any two circuit states in the same subspace differ by at most 1.

[5] The empirical Bayes classifier operates as follows: given observed (and discretized) $d$ components $r^{(d)}(t)$ of the state $r(t)$ it predicts the input which was observed most frequently for the given state components $r^{(d)}(t)$ (maximum a posterior classification, see e.g. [11]). If $r^{(d)}(t)$ was not observed so far a random guess about the input is made.

[6] In fact, in the limit $N \to \infty$ the Bayes classifier is the optimal classifier for the discretized data in the sense that it would yield the lowest classification error — and hence the highest lower bound on mutual information — over all possible classifiers.

[7] In contrast to the Bayes classifier the linear classifiers (both for analog and discrete data) yield already for relatively small sample sizes $N$ good results which do not improve much with increasing $N$.

## References

[1] H. Markram, Y. Wang, and M. Tsodyks. Differential signaling via the same axon of neocortical pyramidal neurons. *Proc. Natl. Acad. Sci.*, 95:5323–5328, 1998.

[2] W. Maass, T. Natschläger, and H. Markram. Real-time computing without stable states: A new framework for neural computation based on perturbations. *Neural Computation*, 14(11):2531–2560, 2002.

[3] A. Gupta, Y. Wang, and H. Markram. Organizing principles for a diversity of GABAergic interneurons and synapses in the neocortex. *Science*, 287:273–278, 2000.

[4] T. M. Cover and J. A. Thomas. *Elements of Information Theory*. Wiley, New York, 1991.

[5] M. S. Roulston. Estimating the errors on measured entropy and mutual information. *Physica D*, 125:285–294, 1999.

[6] S. Panzeri and A. Treves. Analytical estimates of limited sampling biases in different information measures. *Network: Computation in Neural Systems*, 7:87–107, 1996.

[7] G. Pola, S. R. Schultz, R. S. Petersen, and S. Panzeri. A practical guide to information analysis of spike trains. In R. Kötter, editor, *Neuroscience Databases. A Practical Guide*, chapter 10, pages 139–153. Kluwer Academic Publishers (Boston), 2003.

[8] L. Paninski. Estimation of entropy and mutual information. *Neural Computation*, 15:1191–1253, 2003.

[9] J. Hertz. Reading the information in the outcome of neural computation. online available via http://www.nordita.dk/~hertz/papers/infit.ps.gz.

[10] S.P. Strong, R. Koberle, R. R. de Ruyter van Steveninck, and E. Bialek. Entropy and information in neural spike trains. *Physical Review Letters*, 80(1):197–200, 1998.

[11] R. O. Duda, P.E. Hart, and D. G. Stork. *Pattern Classification*. John Wiley & Sons, 2nd edition, 2001.

[12] S. Thorpe, D. Fize, and C. Marlot. Speed of processing in the human visual system. *Nature*, 381:520–522, 1996.
